# Convergence of Large Margin Separable Linear Classification

**Tong Zhang**
Mathematical Sciences Department
IBM T.J. Watson Research Center
Yorktown Heights, NY 10598
tzhang@watson.ibm.com

## Abstract

Large margin linear classification methods have been successfully applied to many applications. For a linearly separable problem, it is known that under appropriate assumptions, the expected misclassification error of the computed "optimal hyperplane" approaches zero at a rate proportional to the inverse training sample size. This rate is usually characterized by the margin and the maximum norm of the input data. In this paper, we argue that another quantity, namely the robustness of the input data distribution, also plays an important role in characterizing the convergence behavior of expected misclassification error. Based on this concept of robustness, we show that for a large margin separable linear classification problem, the expected misclassification error may converge exponentially in the number of training sample size.

## 1 Introduction

We consider the binary classification problem: to determine a label $y \in \{-1, 1\}$ associated with an input vector $x$. A useful method for solving this problem is by using linear discriminant functions. Specifically, we seek a weight vector $w$ and a threshold $\theta$ such that $w^T x < \theta$ if its label $y = -1$ and $w^T x \geq \theta$ if its label $y = 1$.

In this paper, we are mainly interested in problems that are linearly separable by a positive margin (although, as we shall see later, our analysis is suitable for non-separable problems). That is, there exists a hyperplane that perfectly separates the in-class data from the out-of-class data. We shall also assume $\theta = 0$ throughout the rest of the paper for simplicity. This restriction usually does not cause problems in practice since one can always append a constant feature to the input data $x$, which offset the effect of $\theta$.

For linearly separable problems, given a training set of $n$ labeled data $(x^1, y^1), \ldots, (x^n, y^n)$, Vapnik recently proposed a method that optimizes a hard margin bound which he calls the "optimal hyperplane" method (see [11]). The optimal hyperplane $w_n$ is the solution to the following quadratic programming problem:

$$\min_w \frac{1}{2} w^2 \quad \text{s.t.} \quad w^T x^i y^i \geq 1 \quad \text{for } i = 1, \ldots, n. \tag{1}$$

For linearly non-separable problems, a generalization of the optimal hyperplane method has appeared in [2], where a slack variable $\xi^i$ is introduced for each data point $(x^i, y^i)$ for $i = 1, \ldots, n$. We compute a hyperplane $w_n$ that solves

$$\min_{w,\xi} \frac{1}{2} w^T w + C \sum_i \xi^i \quad \text{s.t.} \quad w^T x^i y^i \geq 1 - \xi^i, \quad \xi^i \geq 0 \quad \text{for } i = 1, \ldots, n. \quad (2)$$

Where $C > 0$ is a given parameter (also see [11]).

In this paper, we are interested in the quality of the computed weight $w_n$ for the purpose of predicting the label $y$ of an unseen data point $x$. We study this predictive power of $w_n$ in the standard batch learning framework. That is, we assume that the training data $(x^i, y^i)$ for $i = 1, \ldots n$ are independently drawn from the same underlying data distribution $D$ which is unknown. The predictive power of the computed parameter $w_n$ then corresponds to the classification performance of $w_n$ with respect to the true distribution $D$.

We organize the paper as follows. In Section 2, we briefly review a number of existing techniques for analyzing separable linear classification problems. We then derive an exponential convergence rate of misclassification error in Section 3 for certain large margin linear classification. Section 4 compares the newly derived bound with known results from the traditional margin analysis. We explain that the exponential bound relies on a new quantity (the robustness of the distribution) which is not explored in a traditional margin bound. Note that for certain batch learning problems, exponential learning curves have already been observed [10]. It is thus not surprising that an exponential rate of convergence can be achieved by large margin linear classification.

## 2   Some known results on generalization analysis

There are a number of ways to obtain bounds on the generalization error of a linear classifier. A general framework is to use techniques from empirical processes (aka VC analysis). Many such results that are related to large margin classification have been described in chapter 4 of [3].

The main advantage of this framework is its generality. The analysis does not require the estimated parameter to converge to the true parameter, which is ideal for combinatorial problems. However, for problems that are numerical in natural, the potential parameter space can be significantly reduced by using the first order condition of the optimal solution. In this case, the VC analysis may become suboptimal since it assumes a larger search space than what a typical numerical procedure uses. Generally speaking, for a problem that is linearly separable with a large margin, the expected classification error of the computed hyperplane resulted from this analysis is of the order $O(\frac{\log n}{n})$.[1] Similar generalization bounds can also be obtained for non-separable problems.

In chapter 10 of [11], Vapnik described a leave-one-out cross-validation analysis for linearly separable problems. This analysis takes into account the first order KKT condition of the optimal hyperplane $w_n$. The expected generalization performance from this analysis is $O(\frac{1}{n})$, which is better than the corresponding bounds from the VC analysis. Unfortunately, this technique is only suitable for deriving an expected generalization bound (for example, it is not useful for obtaining a PAC style probability bound).

Another well-known technique for analyzing linearly separable problems is the mistake bound framework in online learning. It is possible to obtain an algorithm with a small generalization error in the batch learning setting from an algorithm with a small online mistake

bound. The readers are referred to [6] and references therein for this type of analysis. The technique may lead to a bound with an expected generalization performance of $O(\frac{1}{n})$.

Besides the above mentioned approaches, generalization ability can also be studied in the statistical mechanical learning framework. It was shown that for linearly separable problems, exponential decrease of misclassification error is possible under this framework [1, 5, 7, 8]. Unfortunately, it is unclear how to relate the statistical mechanical learning framework to the batch learning framework considered in this paper. Their analysis, employing approximation techniques, does not seem to imply small sample bounds which we are interested in.

The statistical mechanical learning result suggests that it may be possible to obtain a similar exponential decay of misclassification error in the batch learning setting, which we prove in the next section. Furthermore, we show that the exponential rate depends on a quantity that is different than the traditional margin concept. Our analysis relies on a PAC style probability estimate on the convergence rate of the estimated parameter from (2) to the true parameter. Consequently, it is suitable for non-separable problems. A direct analysis on the convergence rate of the estimated parameter to the true parameter is important for problems that are numerical in nature such as (2). However, a disadvantage of our analysis is that we are unable to directly deal with the linearly separable formulation (1).

## 3   Exponential convergence

We can rewrite the SVM formulation (2) by eliminating $\xi$ as:

$$w_n(\lambda) = \arg\min_w \frac{1}{n} \sum_i f(w^T x^i y^i - 1) + \frac{\lambda}{2} w^T w, \tag{3}$$

where $\lambda = 1/(nC)$ and

$$f(z) = \begin{cases} -z & z \leq 0, \\ 0 & z > 0. \end{cases}$$

Denote by $D$ the true underlying data distribution of $(x, y)$, and let $w_*(\lambda)$ be the optimal solution with respect to the true distribution as:

$$w_*(\lambda) = \arg\inf_w E_D f(w^T x y - 1) + \frac{\lambda}{2} w^T w. \tag{4}$$

Let $w_*$ be the solution to

$$w_* = \arg\inf_w \frac{1}{2} w^2 \qquad \text{s.t.} \quad E_D f(w^T x y - 1) = 0, \tag{5}$$

which is the infinite-sample version of the optimal hyperplane method.

Throughout this section, we assume $\|w_*\|_2 < \infty$, and $E_D \|x\|_2 < \infty$. The latter condition ensures that $E_D f(w^T x y - 1) \leq \|w\|_2 E_D \|x\|_2 + 1$ exists for all $w$.

### 3.1   Continuity of solution under regularization

In this section, we show that $\|w_*(\lambda) - w_*\|_2 \to 0$ as $\lambda \to 0$. This continuity result allows us to approximate (5) by using (4) and (3) with a small positive regularization parameter $\lambda$. We only need to show that within any sequence of $\lambda$ that converges to zero, there exists a subsequence $\lambda_i \to 0$ such that $w_*(\lambda_i)$ converges to $w_*$ strongly.

We first consider the following inequality which follows from the definition of $w_*(\lambda)$:

$$E_D f(w_*(\lambda)^T x y - 1) + \frac{\lambda}{2} w_*(\lambda)^2 \leq \frac{\lambda}{2} w_*^2. \tag{6}$$

Therefore $\|w_*(\lambda)\|_2 \le \|w_*\|_2$.

It is well-known that every bounded sequence in a Hilbert space contains a weakly convergent subsequence (cf. Proposition 66.4 in [4]). Therefore within any sequence of $\lambda$ that converges to zero, there exists a subsequence $\lambda_i \to 0$ such that $w_*(\lambda_i)$ converges weakly. We denote the limit by $\tilde{w}$.

Since $f(w_*(\lambda)^T xy - 1)$ is dominated by $\|w_*\|_2\|x\|_2 + 1$ which has a finite integral with respect to $D$, therefore from (6) and the Lebesgue dominated convergence theorem, we obtain

$$0 = \lim_i E_D f(w_*(\lambda_i)^T xy - 1) = E_D \lim_i f(w_*(\lambda_i)^T xy - 1) = E_D f(\tilde{w}^T xy - 1). \quad (7)$$

Also note that $\|\tilde{w}\|_2 \le \lim_i \|w_*(\lambda_i)\|_2 \le \|w_*\|_2$, therefore by the definition of $w_*$, we must have $\tilde{w} = w_*$.

Since $w_*$ is the weak limit of $w_*(\lambda_i)$, we obtain $\|w_*\|_2 \le \lim_i \|w_*(\lambda_i)\|_2$. Also since $\|w_*(\lambda_i)\|_2 \le \|w_*\|_2$, therefore $\lim_i \|w_*(\lambda_i)\|_2 = \|w_*\|_2$. This equality implies that $w_*(\lambda_i)$ converges to $w_*$ strongly since

$$\lim_i (w_*(\lambda_i) - w_*)^2 = \lim_i w_*(\lambda_i)^2 + w_*^2 - 2\lim_i w_*(\lambda_i)^T w_* = 0.$$

### 3.2  Accuracy of estimated hyperplane with non-zero regularization parameter

Our goal is to show that for the estimation method (3) with a nonzero regularization parameter $\lambda > 0$, the estimated parameter $w_n(\lambda)$ converges to the true parameter $w_*(\lambda)$ in probability when the sample size $n \to \infty$. Furthermore, we give a large deviation bound on the rate of convergence.

From (4), we obtain the following first order condition:

$$E_D \beta(\lambda, x, y)xy + \lambda w_*(\lambda) = 0, \quad (8)$$

where $\beta(\lambda, x, y) = f'(w_*(\lambda)^T xy - 1)$ and $f'(z) \in [-1, 0]$ denotes a member of the subgradient of $f$ at $z$ [9].[2] In the finite sample case, we can also interpret $\beta(\lambda, x, y)$ in (8) as a scaled dual variable $\alpha$: $\beta = -\alpha/C$, where $\alpha$ appears in the dual (or Kernel) formulation of an SVM (for example, see chapter 10 of [11]).

The convexity of $f$ implies that $f(z_1) + (z_2 - z_1)f'(z_1) \le f(z_2)$ for any subgradient $f'$ of $f$. This implies the following inequality:

$$\frac{1}{n}\sum_i f(w_*(\lambda)^T x^i y^i - 1) + (w_n(\lambda) - w_*(\lambda))^T \frac{1}{n}\sum_i \beta(\lambda, x^i, y^i)x^i y^i$$

$$\le \frac{1}{n}\sum_i f(w_n(\lambda)^T x^i y^i),$$

which is equivalent to:

$$\frac{1}{n}\sum_i f(w_*(\lambda)^T x^i y^i - 1) + \frac{\lambda}{2}w_*(\lambda)^2 +$$

$$(w_n(\lambda) - w_*(\lambda))^T [\frac{1}{n}\sum_i \beta(\lambda, x^i, y^i)x^i y^i + \lambda w_*(\lambda)] + \frac{\lambda}{2}(w_*(\lambda) - w_n(\lambda))^2$$

$$\le \frac{1}{n}\sum_i f(w_n(\lambda)^T x^i y^i - 1) + \frac{\lambda}{2}w_n(\lambda)^2.$$

Also note that by the definition of $w_n(\lambda)$, we have:

$$\frac{1}{n}\sum_i f(w_n(\lambda)^T x^i y^i - 1) + \frac{\lambda}{2}w_n(\lambda)^2 \leq \frac{1}{n}\sum_i f(w_*(\lambda)^T x^i y^i - 1) + \frac{\lambda}{2}w_*(\lambda)^2.$$

Therefore by comparing the above two inequalities, we obtain:

$$\frac{\lambda}{2}(w_*(\lambda) - w_n(\lambda))^2 \leq (w_*(\lambda) - w_n(\lambda))^T [\frac{1}{n}\sum_i \beta(\lambda, x^i, y^i) x^i y^i + \lambda w_*(\lambda)]$$

$$\leq \|w_*(\lambda) - w_n(\lambda)\|_2 \|\frac{1}{n}\sum_i \beta(\lambda, x^i, y^i) x^i y^i + \lambda w_*(\lambda)\|_2.$$

Therefore we have

$$\|w_*(\lambda) - w_n(\lambda)\|_2 \leq \frac{2}{\lambda}\|\frac{1}{n}\sum_i \beta(\lambda, x^i, y^i) x^i y^i + \lambda w_*(\lambda)\|_2$$

$$= \frac{2}{\lambda}\|\frac{1}{n}\sum_i \beta(\lambda, x^i, y^i) x^i y^i - E_D \beta(\lambda, x, y) xy\|_2. \qquad (9)$$

Note that in (9), we have already bounded the convergence of $w_n(\lambda)$ to $w_*(\lambda)$ in terms of the convergence of the empirical expectation of a random vector $\beta(\lambda, x, y)xy$ to its mean. In order to obtain a large deviation bound on the convergence rate, we need the following result which can be found in [13], page 95:

**Theorem 3.1** *Let $\xi_i$ be zero-mean independent random vectors in a Hilbert space. If there exists $M > 0$ such that for all natural numbers $l \geq 2$: $\sum_{i=1}^n E\|\xi_i\|_2^l \leq \frac{nb}{2}l!M^l$. Then for all $\delta > 0$: $P(\|\frac{1}{n}\sum_i \xi_i\|_2 \geq \delta) \leq 2\exp(-\frac{n}{2}\delta^2/(bM^2 + \delta M))$.*

Using the fact that $\beta(\lambda, x, y) \in [-1, 0]$, it is easy to verify the following corollary by using Theorem 3.1 and (9), where we also bound the $l$-th moment of the right hand side of (9) using the following form of Jensen's inequality: $|a + b|^l \leq 2^{l-1}(|a|^l + |b|^l)$ for $l \geq 2$.

**Corollary 3.1** *If there exists $M > 0$ such that for all natural numbers $l \geq 2$: $E_D\|x\|_2^l \leq \frac{b}{2}l!M^l$. Then for all $\delta > 0$:*

$$P(\|w_*(\lambda) - w_n(\lambda)\|_2 \geq \delta) \leq 2\exp(-\frac{n}{8}\lambda^2\delta^2/(4bM^2 + \lambda\delta M)).$$

Let $P_D(\cdot)$ denote the probability with respect to distribution $D$, then the following bound on the expected misclassification error of the computed hyperplane $w_n(\lambda)$ is a straightforward consequence of Corollary 3.1:

**Corollary 3.2** *Under the assumptions of Corollary 3.1, then for any non-random values $\lambda, \gamma, K > 0$, we have:*

$$E_X P_D(w_n(\lambda)^T xy \leq 0) \leq P_D(w_*(\lambda)^T xy \leq \gamma) + P_D(\|x\|_2 \geq K)$$

$$+ 2\exp(-\frac{n}{8}\lambda^2\gamma^2/(4bK^2M^2 + \lambda\gamma KM)),$$

*where the expectation $E_X$ is taken over $n$ random samples from $D$ with $w_n(\lambda)$ estimated from the $n$ samples.*

We now consider linearly separable classification problems where the solution $w_*$ of (5) is finite. Throughout the rest of this section, we impose an additional assumption that the

distribution $D$ is finitely supported: $\|x\|_2 \leq M$ almost everywhere with respect to the measure $D$.

From Section 3.1, we know that for any sufficiently small positive number $\lambda$, $\|w_* - w_*(\lambda)\|_2 < 1/M$, which means that $w_*(\lambda)$ also separates the in-class data from the out-of-class data with a margin of at least $2(1 - M\|w_* - w_*(\lambda)\|_2)$. Therefore for sufficiently small $\lambda$, we can define:

$$\gamma(\lambda) = \sup\{b : P_D(w_*(\lambda)^T xy \leq b) = 0\} \geq 1 - M\|w_* - w_*(\lambda)\|_2 > 0.$$

By Corollary 3.2, we obtain the following upper-bound on the misclassification error if we compute a linear separator from (3) with a non-zero small regularization parameter $\lambda$:

$$E_X P_D(w_n(\lambda)^T xy \leq 0) \leq 2\exp(-\frac{n}{8}\lambda^2\gamma(\lambda)^2/(4M^4 + \lambda\gamma(\lambda)M^2)).$$

This indicates that the expected misclassification error of an appropriately computed hyperplane for a linearly separable problem is exponential in $n$. However, the rate of convergence depends on $\lambda\gamma(\lambda)/M^2$. This quantity is different than the margin concept which has been widely used in the literature to characterize the generalization behavior of a linear classification problem. The new quantity measures the convergence rate of $w_*(\lambda)$ to $w_*$ as $\lambda \to 0$. The faster the convergence, the more "robust" the linear classification problem is, and hence the faster the exponential decay of misclassification error is. As we shall see in the next section, this "robustness" is related to the degree of outliers in the problem.

## 4  Example

We give an example to illustrate the "robustness" concept that characterizes the exponential decay of misclassification error. It is known from Vapnik's cross-validation bound in [11] (Theorem 10.7) that by using the large margin idea alone, one can derive an expected misclassification error bound that is of the order $O(1/n)$, where the constant is margin dependent. We show that this bound is tight by using the following example.

**Example 4.1** Consider a two-dimensional problem. Assume that with probability of $1 - \gamma$, we observe a data point $x$ with label $y$ such that $xy = [1, 0]$; and with probability of $\gamma$, we observe a data point $x$ with label $y$ such that $xy = [-1, 1]$. This problem is obviously linearly separable with a large margin that is $\gamma$ independent.

Now, for $n$ random training data, with probability at most $\gamma^n + (1 - \gamma)^n$, we observe either $x^i y^i = [1, 0]$ for all $i = 1, \ldots, n$, or $x^i y^i = [-1, 1]$ for all $i = 1, \ldots, n$. For all other cases, the computed optimal hyperplane $w_n = w_*$. This means that the misclassification error is $\gamma(1 - \gamma)(\gamma^{n-1} + (1 - \gamma)^{n-1})$. This error converges to zero exponentially as $n \to \infty$. However the convergence rate depends on the fraction of outliers in the distribution characterized by $\gamma$.

In particular, for any $n$, if we let $\gamma = 1/n$, then we have an expected misclassification error that is at least $\frac{1}{n}(1 - 1/n)^n \approx 1/(en)$. $\square$

The above tightness construction of the linear decay rate of the expected generalization error (using the margin concept alone) requires the scenario that a small fraction (which shall be in the order of inverse sample size) of data are very different from other data. This small portion of data can be considered as outliers, which can be measured by the "robustness" of the distribution. In general, $w_*(\lambda)$ converges to $w_*$ slowly when there exist such a small portion of data (outliers) that cannot be correctly classified from the observation of the remaining data. It can be seen that the optimal hyperplane in (1) is quite sensitive to even a single outlier. Intuitively, this instability is quite undesirable. However, the previous large margin learning bounds seemed to have dismissed this concern. This

paper indicates that such a concern is still valid. In the worst case, even if the problem is separable by a large margin, outliers can still cause a slow down of the exponential convergence rate.

## 5 Conclusion

In this paper, we derived new generalization bounds for large margin linearly separable classification. Even though we have only discussed the consequence of this analysis for separable problems, the technique can be easily applied to non separable problems (see Corollary 3.2). For large margin separable problems, we show that exponential decay of generalization error may be achieved with an appropriately chosen regularization parameter. However, the bound depends on a quantity which characterizes the robustness of the distribution. An important difference of the robustness concept and the margin concept is that outliers may not be observable with large probability from data while margin generally will. This implies that without any prior knowledge, it could be difficult to directly apply our bound using only the observed data.

## Footnotes

[1]Bounds described in [3] would imply an expected classification error of $O(\frac{\log^2 n}{n})$, which can be slightly improved (by a $\log n$ factor) if we adopt a slightly better covering number estimate such as the bounds in [12, 14].

[2]For readers not familiar with the subgradient concept in convex analysis, our analysis requires little modification if we replace $f$ with a smoother convex function such as $f^2$, which avoids the discontinuity in the first order derivative.

## References

[1] J.K. Anlauf and M. Biehl. The AdaTron: an adaptive perceptron algorithm. *Europhys. Lett.*, 10(7):687–692, 1989.

[2] C. Cortes and V.N. Vapnik. Support vector networks. *Machine Learning*, 20:273–297, 1995.

[3] Nello Cristianini and John Shawe-Taylor. *An Introduction to Support Vector Machines and other Kernel-based Learning Methods*. Cambridge University Press, 2000.

[4] Harro G. Heuser. *Functional analysis*. John Wiley & Sons Ltd., Chichester, 1982. Translated from the German by John Horváth, A Wiley-Interscience Publication.

[5] W. Kinzel. Statistical mechanics of the perceptron with maximal stability. In *Lecture Notes in Physics*, volume 368, pages 175–188. Springer-Verlag, 1990.

[6] J. Kivinen and M.K. Warmuth. Additive versus exponentiated gradient updates for linear prediction. *Journal of Information and Computation*, 132:1–64, 1997.

[7] M. Opper. Learning times of neural networks: Exact solution for a perceptron algorithm. *Phys. Rev. A*, 38(7):3824–3826, 1988.

[8] M. Opper. Learning in neural networks: Solvable dynamics. *Europhysics Letters*, 8(4):389–392, 1989.

[9] R. Tyrrell Rockafellar. *Convex analysis*. Princeton University Press, Princeton, NJ, 1970.

[10] Dale Schuurmans. Characterizing rational versus exponential learning curves. *J. Comput. Syst. Sci.*, 55:140–160, 1997.

[11] V.N. Vapnik. *Statistical learning theory*. John Wiley & Sons, New York, 1998.

[12] Robert C. Williamson, Alexander J. Smola, and Bernhard Schölkopf. Entropy numbers of linear function classes. In *COLT'00*, pages 309–319, 2000.

[13] Vadim Yurinsky. *Sums and Gaussian vectors*. Springer-Verlag, Berlin, 1995.

[14] Tong Zhang. Analysis of regularized linear functions for classification problems. Technical Report RC-21572, IBM, 1999. Abstract in NIPS'99, pp. 370–376.
